# Lazy Learning Meets the Recursive Least Squares Algorithm

**Mauro Birattari, Gianluca Bontempi, and Hugues Bersini**
Iridia - Université Libre de Bruxelles
Bruxelles, Belgium
{mbiro, gbonte, bersini}@ulb.ac.be

## Abstract

Lazy learning is a memory-based technique that, once a query is received, extracts a prediction interpolating locally the neighboring examples of the query which are considered relevant according to a distance measure. In this paper we propose a data-driven method to select on a query-by-query basis the optimal number of neighbors to be considered for each prediction. As an efficient way to identify and validate local models, the recursive least squares algorithm is introduced in the context of local approximation and lazy learning. Furthermore, beside the *winner-takes-all* strategy for model selection, a local combination of the most promising models is explored. The method proposed is tested on six different datasets and compared with a state-of-the-art approach.

## 1 Introduction

*Lazy learning* (Aha, 1997) postpones all the computation until an explicit request for a prediction is received. The request is fulfilled interpolating locally the examples considered relevant according to a distance measure. Each prediction requires therefore a local modeling procedure that can be seen as composed of a *structural* and of a *parametric* identification. The parametric identification consists in the optimization of the parameters of the local approximator. On the other hand, structural identification involves, among other things, the selection of a family of local approximators, the selection of a metric to evaluate which examples are more relevant, and the selection of the *bandwidth* which indicates the size of the region in which the data are correctly modeled by members of the chosen family of approximators. For a comprehensive tutorial on local learning and for further references see Atkeson *et al.* (1997).

As far as the problem of bandwidth selection is concerned, different approaches exist. The choice of the bandwidth may be performed either based on some *a priori* assumption or on the data themselves. A further sub-classification of data-driven approaches is of interest

here. On the one hand, a constant bandwidth may be used; in this case it is set by a global optimization that minimizes an error criterion over the available dataset. On the other hand, the bandwidth may be selected locally and tailored for each query point.

In the present work, we propose a method that belongs to the latter class of local data-driven approaches. Assuming a given fixed metric and local linear approximators, the method we introduce selects the bandwidth on a query-by-query basis by means of a local leave-one-out cross-validation. The problem of bandwidth selection is reduced to the selection of the number $k$ of neighboring examples which are given a non-zero weight in the local modeling procedure. Each time a prediction is required for a specific query point, a set of local models is identified, each including a different number of neighbors. The generalization ability of each model is then assessed through a local cross-validation procedure. Finally, a prediction is obtained either combining or selecting the different models on the basis of some statistic of their cross-validation errors.

The main reason to favor a query-by-query bandwidth selection is that it allows better adaptation to the local characteristics of the problem at hand. Moreover, this approach is able to handle directly the case in which the database is updated on-line (Bontempi *et al.*, 1997). On the other hand, a globally optimized bandwidth approach would, in principle, require the global optimization to be repeated each time the distribution of the examples changes.

The major contribution of the paper consists in the adoption of the *recursive least squares* algorithm in the context of lazy learning. This is an appealing and efficient solution to the intrinsically incremental problem of identifying and validating a sequence of local linear models centered in the query point, each including a growing number of neighbors. It is worth noticing here that a leave-one-out cross-validation of each model considered does not involve any significant computational overload, since it is obtained though the PRESS statistic (Myers, 1990) which simply uses partial results returned by the recursive least squares algorithm. Schaal and Atkeson (1998) used already the recursive least squares algorithm for the incremental update of a set of local models. In the present paper, we use for the first time this algorithm in a query-by-query perspective as an effective way to explore the neighborhood of each query point.

As a second contribution, we propose a comparison, on a local scale, between a *competitive* and a *cooperative* approach to model selection. On the problem of extracting a final prediction from a set of alternatives, we compared a *winner-takes-all* strategy with a strategy based on the *combination of estimators* (Wolpert, 1992).

In Section 5 an experimental analysis of the recursive algorithm for local identification and validation is presented. The algorithm proposed, used in conjunction with different strategies for model selection or combination, is compared experimentally with Cubist, the rule-based tool developed by Ross Quinlan for generating piecewise-linear models.

## 2   Local Weighted Regression

Given two variables $\mathbf{x} \in \Re^m$ and $y \in \Re$, let us consider the mapping $f\colon \Re^m \to \Re$, known only through a set of $n$ examples $\{(\mathbf{x}_i, y_i)\}_{i=1}^{n}$ obtained as follows:

$$y_i = f(\mathbf{x}_i) + \varepsilon_i, \tag{1}$$

where $\forall i$, $\varepsilon_i$ is a random variable such that $E[\varepsilon_i] = 0$ and $E[\varepsilon_i\varepsilon_j] = 0$, $\forall j \neq i$, and such that $E[\varepsilon_i^m] = \mu_m(\mathbf{x}_i)$, $\forall m \geq 2$, where $\mu_m(\cdot)$ is the unknown $m^{\text{th}}$ moment of the distribution of $\varepsilon_i$ and is defined as a function of $\mathbf{x}_i$. In particular for $m = 2$, the last of the above mentioned properties implies that no assumption of global homoscedasticity is made.

The problem of local regression can be stated as the problem of estimating the value that the regression function $f(\mathbf{x}) = E[y|\mathbf{x}]$ assumes for a specific query point $\mathbf{x}$, using information pertaining only to a neighborhood of $\mathbf{x}$.

Given a query point $\mathbf{x}_q$, and under the hypothesis of a local homoscedasticity of $\varepsilon_i$, the parameter $\beta$ of a local linear approximation of $f(\cdot)$ in a neighborhood of $\mathbf{x}_q$ can be obtained solving the local polynomial regression:

$$\sum_{i=1}^{n} \left\{ (y_i - \mathbf{x}_i'\beta)^2 \, K\left(\frac{d(\mathbf{x}_i, \mathbf{x}_q)}{h}\right) \right\}, \tag{2}$$

where, given a metric on the space $\Re^m$, $d(\mathbf{x}_i, \mathbf{x}_q)$ is the distance from the query point to the $i^{\text{th}}$ example, $K(\cdot)$ is a weight function, $h$ is the bandwidth, and where a constant value 1 has been appended to each vector $\mathbf{x}_i$ in order to consider a constant term in the regression.

In matrix notation, the solution of the above stated weighted least squares problem is given by:

$$\hat{\beta} = (\mathbf{X}'\mathbf{W}'\mathbf{W}\mathbf{X})^{-1}\mathbf{X}'\mathbf{W}'\mathbf{W}\mathbf{y} = (\mathbf{Z}'\mathbf{Z})^{-1}\mathbf{Z}'\mathbf{v} = \mathbf{P}\mathbf{Z}'\mathbf{v}, \tag{3}$$

where $\mathbf{X}$ is a matrix whose $i^{\text{th}}$ row is $\mathbf{x}_i'$, $\mathbf{y}$ is a vector whose $i^{\text{th}}$ element is $y_i$, $\mathbf{W}$ is a diagonal matrix whose $i^{\text{th}}$ diagonal element is $w_{ii} = \sqrt{K\left(d(\mathbf{x}_i, \mathbf{x}_q)/h\right)}$, $\mathbf{Z} = \mathbf{W}\mathbf{X}$, $\mathbf{v} = \mathbf{W}\mathbf{y}$, and the matrix $\mathbf{X}'\mathbf{W}'\mathbf{W}\mathbf{X} = \mathbf{Z}'\mathbf{Z}$ is assumed to be non-singular so that its inverse $\mathbf{P} = (\mathbf{Z}'\mathbf{Z})^{-1}$ is defined.

Once obtained the local linear polynomial approximation, a prediction of $y_q = f(\mathbf{x}_q)$, is finally given by:

$$\hat{y}_q = \mathbf{x}_q'\hat{\beta}. \tag{4}$$

Moreover, exploiting the linearity of the local approximator, a leave-one-out cross-validation estimation of the error variance $E[(y_q - \hat{y}_q)^2]$ can be obtained without any significant overload. In fact, using the PRESS statistic (Myers, 1990), it is possible to calculate the error $e_j^{\text{cv}} = y_j - \mathbf{x}_j'\hat{\beta}_{-j}$, without explicitly identifying the parameters $\hat{\beta}_{-j}$ from the examples available with the $j^{\text{th}}$ removed. The formulation of the PRESS statistic for the case at hand is the following:

$$e_j^{\text{cv}} = y_j - \mathbf{x}_j'\hat{\beta}_{-j} = \frac{y_j - \mathbf{x}_j'\mathbf{P}\mathbf{Z}'\mathbf{v}}{1 - \mathbf{z}_j'\mathbf{P}\mathbf{z}_j} = \frac{y_j - \mathbf{x}_j'\hat{\beta}}{1 - h_{jj}}, \tag{5}$$

where $\mathbf{z}_j'$ is the $j^{\text{th}}$ row of $\mathbf{Z}$ and therefore $\mathbf{z}_j = w_{jj}\mathbf{x}_j$, and where $h_{jj}$ is the $j^{\text{th}}$ diagonal element of the *Hat matrix* $\mathbf{H} = \mathbf{Z}\mathbf{P}\mathbf{Z}' = \mathbf{Z}(\mathbf{Z}'\mathbf{Z})^{-1}\mathbf{Z}'$.

## 3 Recursive Local Regression

In what follows, for the sake of simplicity, we will focus on linear approximator. An extension to generic polynomial approximators of any degree is straightforward. We will assume also that a metric on the space $\Re^m$ is given. All the attention will be thus centered on the problem of bandwidth selection.

If as a weight function $K(\cdot)$ the indicator function

$$K\left(\frac{d(\mathbf{x}_i, \mathbf{x}_q)}{h}\right) = \begin{cases} 1 & \text{if } d(\mathbf{x}_i, \mathbf{x}_q) \leq h, \\ 0 & \text{otherwise;} \end{cases} \tag{6}$$

is adopted, the optimization of the parameter $h$ can be conveniently reduced to the optimization of the number $k$ of neighbors to which a unitary weight is assigned in the local

regression evaluation. In other words, we reduce the problem of bandwidth selection to a search in the space of $h(k) = d(\mathbf{x}(k), \mathbf{x}_q)$, where $\mathbf{x}(k)$ is the $k^{\text{th}}$ nearest neighbor of the query point.

The main advantage deriving from the adoption of the weight function defined in Eq. 6, is that, simply by updating the parameter $\hat{\beta}(k)$ of the model identified using the $k$ nearest neighbors, it is straightforward and inexpensive to obtain $\hat{\beta}(k + 1)$. In fact, performing a step of the standard recursive least squares algorithm (Bierman, 1977), we have:

$$
\begin{cases}
\mathbf{P}(k+1) = \mathbf{P}(k) - \dfrac{\mathbf{P}(k)\mathbf{x}(k+1)\mathbf{x}'(k+1)\mathbf{P}(k)}{1 + \mathbf{x}'(k+1)\mathbf{P}(k)\mathbf{x}(k+1)} \\[2mm]
\gamma(k+1) = \mathbf{P}(k+1)\mathbf{x}(k+1) \\[2mm]
e(k+1) = y(k+1) - \mathbf{x}'(k+1)\hat{\beta}(k) \\[2mm]
\hat{\beta}(k+1) = \hat{\beta}(k) + \gamma(k+1)e(k+1)
\end{cases} \tag{7}
$$

where $\mathbf{P}(k) = (\mathbf{Z}'\mathbf{Z})^{-1}$ when $h = h(k)$, and where $\mathbf{x}(k + 1)$ is the $(k + 1)^{\text{th}}$ nearest neighbor of the query point.

Moreover, once the matrix $\mathbf{P}(k + 1)$ is available, the leave-one-out cross-validation errors can be directly calculated without the need of any further model identification:

$$
e_j^{\text{cv}}(k+1) = \frac{y_j - \mathbf{x}_j'\hat{\beta}(k+1)}{1 - \mathbf{x}_j'\mathbf{P}(k+1)\mathbf{x}_j}, \qquad \forall j : \ d(\mathbf{x}_j, \mathbf{x}_q) \leq h(k+1). \tag{8}
$$

It will be useful in the following to define for each value of $k$ the $[k \times 1]$ vector $\mathbf{e}^{\text{cv}}(k)$ that contains all the leave-one-out errors associated to the model $\hat{\beta}(k)$.

Once an initialization $\hat{\beta}(0) = \bar{\beta}$ and $\mathbf{P}(0) = \tilde{\mathbf{P}}$ is given, Eq. 7 and Eq. 8 recursively evaluate for different values of $k$ a local approximation of the regression function $f(\cdot)$, a prediction of the value of the regression function in the query point, and the vector of leave-one-out errors from which it is possible to extract an estimate of the variance of the prediction error. Notice that $\bar{\beta}$ is an *a priori* estimate of the parameter and $\tilde{\mathbf{P}}$ is the covariance matrix that reflects the reliability of $\bar{\beta}$ (Bierman, 1977). For non-reliable initialization, the following is usually adopted: $\tilde{\mathbf{P}} = \lambda \mathbf{I}$, with $\lambda$ large and where $\mathbf{I}$ is the identity matrix.

## 4   Local Model Selection and Combination

The recursive algorithm described by Eq. 7 and Eq. 8 returns for a given query point $\mathbf{x}_q$, a set of predictions $\hat{y}_q(k) = \mathbf{x}_q'\hat{\beta}(k)$, together with a set of associated leave-one-out error vectors $\mathbf{e}^{\text{cv}}(k)$.

From the information available, a final prediction $\hat{y}_q$ of the value of the regression function can be obtained in different ways. Two main paradigms deserve to be considered: the first is based on the selection of the *best* approximator according to a given criterion, while the second returns a prediction as a combination of more local models.

If the selection paradigm, frequently called *winner-takes-all*, is adopted, the most natural way to extract a final prediction $\hat{y}_q$, consists in comparing the prediction obtained for each value of $k$ on the basis of the classical *mean square error* criterion:

$$
\hat{y}_q = \mathbf{x}_q'\hat{\beta}(\hat{k}), \quad \text{with } \hat{k} = \arg\min_k \text{MSE}(k) = \arg\min_k \frac{\sum_{i=1}^k \omega_i \left(\mathbf{e}_i^{\text{cv}}(k)\right)^2}{\sum_{i=1}^k \omega_i}; \tag{9}
$$

Table 1: A summary of the characteristics of the datasets considered.

| Dataset | Housing | Cpu | Prices | Mpg | Servo | Ozone |
|---|---|---|---|---|---|---|
| Number of examples | 506 | 209 | 159 | 392 | 167 | 330 |
| Number of regressors | 13 | 6 | 16 | 7 | 8 | 8 |

where $\omega_i$ are weights than can be conveniently used to discount each error according to the distance from the query point to the point to which the error corresponds (Atkeson *et al.*, 1997).

As an alternative to the *winner-takes-all* paradigm, we explored also the effectiveness of local combinations of estimates (Wolpert, 1992). Adopting also in this case the *mean square error* criterion, the final prediction of the value $y_q$ is obtained as a weighted average of the best $b$ models, where $b$ is a parameter of the algorithm. Suppose the predictions $\hat{y}_q(k)$ and the error vectors $e^{cv}(k)$ have been ordered creating a sequence of integers $\{k_i\}$ so that $\mathrm{MSE}(k_i) \leq \mathrm{MSE}(k_j), \forall i < j$. The prediction of $\hat{y}_q$ is given by

$$\hat{y}_q = \frac{\sum_{i=1}^b \zeta_i \hat{y}_q(k_i)}{\sum_{i=1}^b \zeta_i}, \tag{10}$$

where the weights are the inverse of the mean square errors: $\zeta_i = 1/\mathrm{MSE}(k_i)$. This is an example of the *generalized ensemble method* (Perrone & Cooper, 1993).

## 5   Experiments and Results

The experimental evaluation of the incremental local identification and validation algorithm was performed on six datasets. The first five, described by Quinlan (1993), were obtained from the UCI Repository of machine learning databases (Merz & Murphy, 1998), while the last one was provided by Leo Breiman. A summary of the characteristics of each dataset is presented in Table 1.

The methods compared adopt the recursive identification and validation algorithm, combined with different strategies for model selection or combination. We considered also two approaches in which $k$ is selected globally:

**lb1:** Local bandwidth selection for linear local models. The number of neighbors is selected on a query-by-query basis and the prediction returned is the one of the best model according to the mean square error criterion.

**lb0:** Local bandwidth selection for constant local models. The algorithm for constant models is derived directly from the recursive method described in Eq. 7 and Eq. 8. The best model is selected according to the mean square error criterion.

**lbC:** Local combination of estimators. This is an example of the method described in Eq. 10. On the datasets proposed, for each query the best 2 linear local models and the best 2 constant models are combined.

**gb1:** Global bandwidth selection for linear local models. The value of $k$ is obtained minimizing the prediction error in 20-fold cross-validation on the dataset available. This value is then used for all the query points.

**gb0:** Global bandwidth selection for constant local models. As in **gb1**, the value of $k$ is optimized globally and kept constant for all the queries.

Table 2: Mean absolute error on unseen cases.

| Method | Housing | Cpu | Prices | Mpg | Servo | Ozone |
|---|---|---|---|---|---|---|
| lb1 | 2.21 | 28.38 | 1509 | 1.94 | 0.48 | 3.52 |
| lb0 | 2.60 | 31.54 | 1627 | 1.97 | 0.32 | 3.33 |
| lbC | 2.12 | 26.79 | 1488 | 1.83 | 0.29 | 3.31 |
| gb1 | 2.30 | 28.69 | 1492 | 1.92 | 0.52 | 3.46 |
| gb0 | 2.59 | 32.19 | 1639 | 1.99 | 0.34 | 3.19 |
| Cubist | 2.17 | 28.37 | 1331 | 1.90 | 0.36 | 3.15 |

Table 3: Relative error (%) on unseen cases.

| Method | Housing | Cpu | Prices | Mpg | Servo | Ozone |
|---|---|---|---|---|---|---|
| lb1 | 12.63 | 9.20 | 15.87 | 12.65 | 28.66 | 35.25 |
| lb0 | 18.06 | 20.37 | 22.19 | 12.64 | 22.04 | 31.11 |
| lbC | 12.35 | 9.29 | 17.62 | 11.82 | 19.72 | 30.28 |
| gb1 | 13.47 | 9.93 | 15.95 | 12.83 | 30.46 | 32.58 |
| gb0 | 17.99 | 21.43 | 22.29 | 13.48 | 24.30 | 28.21 |
| Cubist | 16.02 | 12.71 | 11.67 | 12.57 | 18.53 | 26.59 |

As far as the metric is concerned, we adopted a global Euclidean metric based on the relative influence (*relevance*) of the regressors (Friedman, 1994). We are confident that the adoption of a local metric could improve the performance of our lazy learning method.

The results of the methods introduced are compared with those we obtained, in the same experimental settings, with Cubist, the rule-based tool developed by Quinlan for generating piecewise-linear models. Each approach was tested on each dataset using the same 10-fold cross-validation strategy. Each dataset was divided randomly into 10 groups of nearly equal size. In turn, each of these groups was used as a testing set while the remaining ones together were providing the examples. Thus all the methods performed a prediction on the same unseen cases, using for each of them the same set of examples. In Table 2 we present the results obtained by all the methods, and averaged on the 10 cross-validation groups. Since the methods were compared on the same examples in exactly the same conditions, the sensitive one-tailed paired test of significance can be used. In what follows, by "significantly better" we mean better at least at a 5% significance level.

The first consideration about the results concerns the local combination of estimators. According to Table 2, the method **lbC** performs in average always better than the *winner-takes-all* linear and constant. On two dataset **lbC** is significantly better than both **lb1** and **lb0**; and on three dataset it is significantly better than one of the two, and better in average than the other.

The second consideration is about the comparison between our query-by-query bandwidth selection and a global optimization of the number of neighbors: in average **lb1** and **lb0** performs better than their counterparts **gb1** and **gb0**. On two datasets **lb1** is significantly better than **gb1**, while is about the same on the other four. On one dataset **lb0** is significantly better than **gb0**.

As far as the comparison with Cubist is concerned, the recursive lazy identification and validation proposed obtains results comparable with those obtained by the state-of-the-art method implemented in Cubist. On the six datasets, **lbC** performs one time significantly better than Cubist, and one time significantly worse.

The second index of performance we investigated is the *relative error*, defined as the mean square error on unseen cases, normalized by the variance of the test set. The relative errors are presented in Table 3 and show a similar picture to Table 2, although the mean square errors considered here penalize larger absolute errors.

## 6 Conclusion and Future Work

The experimental results confirm that the recursive least squares algorithm can be effectively used in a local context. Despite the trivial metric adopted, the local combination of estimators, identified and validated recursively, showed to be able to compete with a state-of-the-art approach.

Future work will focus on the problem of local metric selection. Moreover, we will explore more sophisticated ways to combine local estimators and we will extend this work to polynomial approximators of higher degree.

## Acknowledgments

The work of Mauro Birattari was supported by the FIRST program of the Région Wallonne, Belgium. The work of Gianluca Bontempi was supported by the European Union TMR Grant FMBICT960692. The authors thank Ross Quinlan and gratefully acknowledge using his software Cubist. For more details on Cubist see http://www.rulequest.com. We also thank Leo Breiman for the dataset *ozone* and the UCI Repository for the other datasets used in this paper.

## References

Aha D. W. 1997. Editorial. *Artificial Intelligence Review*, **11**(1–5), 1–6. Special Issue on Lazy Learning.

Atkeson C. G. , Moore A. W. & Schaal S. 1997. Locally weighted learning. *Artificial Intelligence Review*, **11**(1–5), 11–73.

Bierman G. J. 1977. *Factorization Methods for Discrete Sequential Estimation*. New York, NY: Academic Press.

Bontempi G. , Birattari M. & Bersini H. 1997. Lazy learning for local modeling and control design. *International Journal of Control*. Accepted for publication.

Friedman J. H. 1994. *Flexible metric nearest neighbor classification*. Tech. rept. Department of Statistics, Stanford University.

Merz C. J. & Murphy P. M. 1998. *UCI Repository of machine learning databases*.

Myers R. H. 1990. *Classical and Modern Regression with Applications*. Boston, MA: PWS-KENT.

Perrone M. P. & Cooper L. N. 1993. When networks disagree: Ensemble methods for hybrid neural networks. *Pages 126–142 of:* Mammone R. J. (ed), *Artificial Neural Networks for Speech and Vision*. Chapman and Hall.

Quinlan J. R. 1993. Combining instance-based and model-based learning. *Pages 236–243 of: Machine Learning. Proceedings of the Tenth International Conference*. Morgan Kaufmann.

Schaal S. & Atkeson C. G. 1998. Constructive incremental learning from only local information. *Neural Computation*, **10**(8), 2047–2084.

Wolpert D. 1992. Stacked Generalization. *Neural Networks*, **5**, 241–259.